# Template-Based Algorithms for Connectionist Rule Extraction

**Jay A. Alexander and Michael C. Mozer**
Department of Computer Science and
Institute for Cognitive Science
University of Colorado
Boulder, CO 80309–0430

## Abstract

Casting neural network weights in symbolic terms is crucial for interpreting and explaining the behavior of a network. Additionally, in some domains, a symbolic description may lead to more robust generalization. We present a principled approach to symbolic rule extraction based on the notion of *weight templates*, parameterized regions of weight space corresponding to specific symbolic expressions. With an appropriate choice of representation, we show how template parameters may be efficiently identified and instantiated to yield the optimal match to a unit's actual weights. Depending on the requirements of the application domain, our method can accommodate arbitrary disjunctions and conjunctions with $O(k)$ complexity, simple $n$-of-$m$ expressions with $O(k^2)$ complexity, or a more general class of recursive $n$-of-$m$ expressions with $O(k^3)$ complexity, where $k$ is the number of inputs to a unit. Our method of rule extraction offers several benefits over alternative approaches in the literature, and simulation results on a variety of problems demonstrate its effectiveness.

## 1 INTRODUCTION

The problem of understanding why a trained neural network makes a given decision has a long history in the field of connectionist modeling. One promising approach to this problem is to convert each unit's weights and/or activities from continuous numerical quantities into discrete, symbolic descriptions [2, 4, 8]. This type of reformulation, or *rule extraction*, can both explain network behavior and facilitate transfer of learning. Additionally, in intrinsically symbolic domains, there is evidence that a symbolic description can lead to more robust generalization [4].

We are interested in extracting symbolic rules on a unit-by-unit basis from connectionist nets that employ the conventional inner product activation and sigmoidal output functions. The basic language of description for our rules is that of $n$-of-$m$ expressions. An $n$-of-$m$ expression consists of a list of $m$ subexpressions and a value $n$ such that $1 \le n \le m$. The overall expression is true when at least $n$ of the $m$ subexpressions are true. An example of an $n$-of-$m$ expression stated using logical variables is the majority voter function $X = 2 \, of \, (A, B, C)$. $N$-of-$m$ expressions are interesting because they are able to model behaviors intermediate to standard Boolean OR ($n = 1$) and AND ($n = m$) functions. These intermediate behaviors reflect a limited form of two-level Boolean logic. (To see why this is true, note that the expression for $X$ above is equivalent to $AB + BC + AC$.) In a later section we describe even more general behaviors that can be represented using recursive forms of these expressions. $N$-of-$m$ expressions fit well with the activation behavior of sigmoidal units, and they are quite amenable to human comprehension.

To extract an $n$-of-$m$ rule from a unit's weights, we follow a three-step process. First we generate a minimal set of candidate templates, where each template is parameterized to represent a given $n$-of-$m$ expression. Next we instantiate each template's parameters with optimal values. Finally we choose the symbolic expression whose instantiated template is nearest to the actual weights. Details on each of these steps are given below.

## 2   TEMPLATE-BASED RULE EXTRACTION

### 2.1  Background

Following McMillan [4], we define a *weight template* as a parameterized region of weight space corresponding to a specific symbolic function. To see how weight templates can be used to represent symbolic functions, consider the weight vector for a sigmoidal unit with four inputs and a bias:

$$w \quad = \quad w_1 \quad w_2 \quad w_3 \quad w_4 \quad b$$

Now consider the following two template vectors:

$$t_1 \quad = \quad -p \quad p \quad 0 \quad -p \quad 1.5p$$
$$t_2 \quad = \quad p \quad -p \quad p \quad p \quad -0.5p$$

These templates are parameterized by the variable $p$. Given a large positive value of $p$ (say 5.0) and an input vector $I$ (whose components are approximately 0 and 1), $t_1$ describes the symbolic expression $1 \, of \, (\bar{I}_1, I_2, \bar{I}_4)$, while $t_2$ describes the symbolic expression $2 \, of \, (I_1, \bar{I}_2, I_3, I_4)$. A general description for $n$-of-$m$ templates of this form is the following:

1. $M$ of the weight values are set to $\pm p$, $p > 0$; all others are set to 0.
   ($+p$ is used for normal subexpressions, $-p$ for negated subexpressions)

2. The bias value is set to $(0.5 + m_{neg} - n)p$, where $m_{neg}$ represents the number of negated subexpressions.

When the inputs are Boolean with values $-1$ and $+1$, the form of the templates is the same, except the template bias takes the value $(1 + m - 2n)p$. This seemingly trivial difference turns out to have a significant effect on the efficiency of the extraction process.

## 2.2 Basic extraction algorithm

### Generating candidate templates

Given a sigmoidal unit with $k$ inputs plus a bias, the total number of $n$-of-$m$ expressions that unit may compute is an exponential function of $k$:

$$T = \sum_{m=1}^{k} \sum_{n=1}^{m} 2^m \binom{k}{m} = \sum_{m=1}^{k} \frac{2^m k!}{(k-m)!\,(m-1)!} = 2k3^{k-1}$$

For example, $T_{k=10}$ is 393,660, while $T_{k=20}$ is over 46 *billion*. Fortunately we can apply knowledge of the unit's actual weights to explore this search space without generating a template for each possible $n$-of-$m$ expression. Alexander [1] proves that when the $-1/+1$ input representation is used, we need consider at most one template for each possible choice of $n$ and $m$. For a given choice of $n$ and $m$, a template is indicated when $sign(1 + m - 2n) = sign(b)$. A required template is formed by setting the template weights corresponding to the $m$ highest absolute value actual weights to $sp$, where $s$ represents the sign of the corresponding actual weight. The template bias is set to $(1 + m - 2n)p$. This reduces the number of templates required to a polynomial function of $k$:

$$T_{n-of-m} = \sum_{m=1}^{k} \sum_{n=1}^{\left\lfloor \frac{m+1}{2} \right\rfloor} 1 = \left\lfloor \frac{1}{4}k^2 + \frac{1}{2}k + \frac{1}{4} \right\rfloor$$

Values for $T_{k=10}$ and $T_{k=20}$ are now 30 and 110, respectively, making for a very efficient pruning of the search space. When 0/1 inputs are used, this simple procedure does not suffice and many more templates must be generated. For this reason, in the remainder of this paper we focus on the $-1/+1$ case and assume the use of symmetric sigmoid functions.

### Instantiating template parameters

Instantiating a weight template $t$ requires finding a value for $p$ such that the Euclidean distance $d = \|t - w\|^2$ is minimized. Letting $u_i = 1$ if template weight $t_i$ is nonzero, $u_i = 0$ otherwise, the value of $p$ that minimizes this distance for any $-1/+1$ template is given by:

$$p* = \frac{\sum_{i=1}^{k} |w_i| u_i + (1 + m - 2n)\,b}{m + (1 + m - 2n)^2}$$

### Finding the nearest template and checking extraction validity

Once each template is instantiated with its value of $p*$, the distance between the template and the actual weight vector is calculated, and the minimal distance template is selected as the basis for rule extraction. Having found the nearest template $t*$, we can use its values as part of a rudimentary check on extraction validity. For example, we can define the *extraction error* as $100\% \times \|t* - w\|^2 / \|w\|^2$ to measure how well the nearest symbolic rule fits the actual weights. We can also examine the value of $p*$ used in $t*$. Small values of $p*$ translate into activation levels in the linear regime of the sigmoid functions, compromising the assumption of Boolean outputs propagating to subsequent inputs.

## 2.3   Extending expressiveness

While the $n$-of-$m$ expressions treated thus far are fairly powerful, there is an interesting class of symbolic behaviors that cannot be captured by simple $n$-of-$m$ expressions. The simplest example of this type of behavior may be seen in the single hidden unit version of *xor* described in [6]. In this 2–1–1 network the hidden unit $H$ learns the expression $AND(\bar{I}_1, \bar{I}_2)$, while the output unit (which connects to the two inputs as well as to the hidden unit) learns the expression $AND[OR(\bar{I}_1, \bar{I}_2), \bar{H}]$. This latter expression may be viewed as a nested or recursive form of $n$-of-$m$ expression, one where some of the $m$ subexpressions may themselves be $n$-of-$m$ expressions. The following two forms of recursive $n$-of-$m$ expressions are linearly separable and are thus computable by a single sigmoidal unit:

$$OR\ [\ C_{n\text{-of-}m},\ C_{OR}\ ]$$
$$AND\ [\ C_{n\text{-of-}m},\ C_{AND}\ ]$$

where $C_{n\text{-of-}m}$ is a nested $n$-of-$m$ expression $(1 \le n \le m)$
$\quad C_{OR}\quad$ is a nested $OR$ expression $(n = 1)$
$\quad C_{AND}\quad$ is a nested $AND$ expression $(n = m)$

These expressions may be seen to generalize simple $n$-of-$m$ expressions in the same way that simple $n$-of-$m$ expressions generalize basic disjunctions and conjunctions.[1] We term the above forms *augmented* $n$-of-$m$ expressions because they extend simple $n$-of-$m$ expressions with additional disjuncts or conjuncts. Templates for these expressions (under the $-1/+1$ input representation) may be efficiently generated and instantiated using a procedure similar to that described for simple $n$-of-$m$ expressions. When augmented expressions are included in the search, the total number of templates required becomes:

$$T_{augmented}\ =\ \left\lfloor \frac{1}{6}k^3 - \frac{1}{4}k^2 + \frac{5}{6}k + \frac{1}{4} \right\rfloor$$

This figure is $O(k)$ worse than for simple $n$-of-$m$ expressions, but it is still polynomial in $k$ and is quite manageable for many problems. (Values for $T_{k=10}$ and $T_{k=20}$ are 150 and 1250, respectively.) A more detailed treatment of augmented $n$-of-$m$ expressions is given in [1].

## 3   RELATED WORK

Here we briefly consider two alternative systems for connectionist rule extraction. Many other methods have been developed; a recent summary and categorization appears in [2].

### 3.1   McMillan

McMillan described the *projection* of actual weights to simple weight templates in [4]. McMillan's parameter selection and instantiation procedures are inefficient compared to those described here, though they yield equivalent results for the classes of templates he used. McMillan treated only expressions with $m \le 2$ and no negated subexpressions.

## 3.2 Towell and Shavlik

Towell and Shavlik [8] use a domain theory to initialize a connectionist network, train the network on a set of labeled examples, and then extract rules that describe the network's behavior. To perform rule extraction, Towell and Shavlik first group weights using an iterative clustering algorithm. After applying additional training, they typically check each training pattern against each weight group and eliminate groups that do not affect the classification of any pattern. Finally, they scan remaining groups and attempt to express a rule in purely symbolic *n*-of-*m* form. However, in many cases the extracted rules take the form of a linear inequality involving multiple numeric quantities. For example, the following rule was extracted from part of a network trained on the promoter recognition task [5] from molecular biology:

```
Minus35 = -10 < + 5.0 * nt(@-37 '--T-G--A')
                + 3.1 * nt(@-37 '---GT---')
                + 1.9 * nt(@-37 '----C-CT')
                + 1.5 * nt(@-37 '---C--A-')
                - 1.5 * nt(@-37 '------GC')
                - 1.9 * nt(@-37 '--CAW---')
                - 3.1 * nt(@-37 '--A----C')

where nt() returns the number of true subexpressions,
@-37 locates the subexpressions on the DNA strand,
and "-" indicates a don't-care subexpression.
```

Towell and Shavlik's method can be expected to give more accurate results than our approach, but at a cost. Their method is very compute intensive and relies substantially on access to a fixed set of training patterns. Additionally, it is not clear that their rules are completely symbolic. While numeric expressions were convenient for the domains they studied, in applications where one is interested in more abstract descriptions, such expressions may be viewed as providing too much detail, and may be difficult for people to interpret and reason about. Sometimes one wants to determine the nearest symbolic interpretation of unit behavior rather than a precise mathematical description. Our method offers a simpler paradigm for doing this. Given these differences, we conclude that both methods have their place in rule extraction tool kits.

## 4 SIMULATIONS

### 4.1 Simple logic problems

We used a group of simple logic problems to verify that our extraction algorithms could produce a correct set of rules for networks trained on the complete pattern space of each function. Table 1 summarizes the results.[2] The *rule-plus-exception* problem is defined as $f = AB + \overline{A}\,\overline{B}\,\overline{C}\,\overline{D}$; *xor-1* is the 2–1–1 version of *xor* described in Section 2.3; and *xor-2* is a strictly layered (2–2–1) version of *xor* [6]. The *negation* problem is also described in [6]; in this problem one of the four inputs controls whether the other inputs appear normally or negated at the outputs. (As with *xor-1*, the network for *negation* makes use of direct input/ output connections.) In addition to the perfect classification performance of the rules, the large values of $p*$ and small values of extraction error (as defined in Section 2.2) provide evidence that the extraction process is very accurate.

| Problem | Network Topology | Hidden Unit Penalty Term | Average $p^*$ | | Extraction Error | | Patterns Correctiy Classified by Rules |
|---|---|---|---|---|---|---|---|
| | | | Hidden Unit(s) | Output Unit(s) | Hidden Unit(s) | Output Unit(s) | |
| rule-plus-exception | 4–2–1 | – | 2.72 | 6.15 | 0.8 % | 1.3 % | 100.0 % |
| xor-1 | 2–1–1 | – | 5.68 | 4.40 | 0.1 % | 0.1 % | 100.0 % |
| xor-2 | 2–2–1 | – | 4.34 | 5.68 | 0.4 % | 1.0 % | 100.0 % |
| negation | 4–3–4 | activation | 5.40 | 5.17 | 0.2 % | 2.2 % | 100.0 % |

**Table 1: Simulation summary for simple logic problems**

Symbolic solutions for these problems often come in forms different from the canonical form of the function. For example, the following rules for the *rule-plus-exception* problem show a level of negation within the network:

$$H_1 = OR (A, B, C, D)$$
$$H_2 = AND (A, B)$$
$$O = OR (\overline{H}_1, H_2)$$

Example results on *xor-1* show the expected use of an augmented *n*-of-*m* expression:

$$H = OR (I_1, \overline{I}_2)$$
$$O = OR [AND(I_1, \overline{I}_2), \overline{H}]$$

## 4.2 The MONK's problems

We tested generalization performance using the MONK's problems [5, 7], a set of three problems used to compare a variety of symbolic and connectionist learning algorithms. A summary of these tests appears in Table 2. Our performance was equal to or better than all of the systems tested in [7] for the *monks-1* and *monks-2* problems. Moreover, the rules extracted by our algorithm were very concise and easy to understand, in contrast to those produced by several of the symbolic systems. (The two connectionist systems reported in [7] were opaque, i.e., no rules were extracted.) As an example, consider the following output for the *monks-2* problem:

```
H₁ = 2 of (head_shape round, body_shape round, is_smiling yes,
           holding sword, jacket_color red, has_tie yes)
H₂ = 3 of (head_shape round, body_shape round, is_smiling yes,
           holding sword, jacket_color red, has_tie not no)
O  = AND (H₁, H̄₂)
```

The target concept for this problem is *exactly 2 of the attributes have their first value*. These rules demonstrate an elegant use of *n*-of-*m* expressions to describe the idea of "exactly 2" as "at least 2 but not 3". The *monks-3* problem is difficult due to (intentional) training set noise, but our results are comparable to the other systems tested in [7].

| Problem | Network Topology | Hidden Unit Penalty Term | Training Set | | | Test Set | | |
|---|---|---|---|---|---|---|---|---|
| | | | # of Patterns | Perf. of Network | Perf. of Rules | # of Patterns | Perf. of Network | Perf. of Rules |
| monks-1 | 17–3–1 | decay | 124 | 100.0 % | 100.0 % | 432 | 100.0 % | 100.0 % |
| monks-2 | 17–2–1 | decay | 169 | 100.0 % | 100.0 % | 432 | 100.0 % | 100.0 % |
| monks-3 | 17–0–1 | – | 122 | 93.4 % | 93.4 % | 432 | 97.2 % | 97.2 % |

Table 2: Simulation summary for the MONK's problems

## 4.3 UCI repository problems

The final set of simulations addresses extraction performance on three real-world databases from the UCI repository [5]. Table 3 shows that good results were achieved. For the *promoters* task, we achieved generalization performance of nearly 88%, compared to 93-96% reported by Towell and Shavlik [8]. However, our results are impressive when viewed in light of the simplicity and comprehensibility of the extracted output. While Towell and Shavlik's results for this task included 5 rules like the one shown in Section 3.2, our single rule is quite simple:

```
promoter  =  5 of (@-45 'AA-------TTGA-A-----T------T-----AAA----C')
```

Results for the *house-votes-84* and *breast-cancer-wisc* problems are especially noteworthy since the generalization performance of the rules is virtually identical to that of the raw networks. This indicates that the rules are capturing a significant portion of the computation being performed by the networks. The following rule was the one most frequently extracted for the *house-votes-84* problem, where the task is to predict party affiliation:

```
Democrat  =  OR [ 5 of (V₃, V̄₇, V₉, V₁₀, V₁₁, V̄₁₂), V̄₄ ]

where  V₃  = voted for adoption-of-the-budget-resolution bill
       V₄  = voted for physician-fee-freeze bill
       V₇  = voted for anti-satellite-test-ban bill
       V₉  = voted for mx-missile bill
       V₁₀ = voted for immigration bill
       V₁₁ = voted for synfuels-corporation-cutback bill
       V₁₂ = voted for education-spending bill
```

Shown below is a typical rule set extracted for the *breast-cancer-wisc* problem. Here the goal is to diagnose a tumor as benign or malignant based on nine clinical attributes.

```
Malignant = AND (H₁, H₂)
       H₁ = 4 of (thickness > 3, size > 1, adhesion > 1, epithelial > 5,
                  nuclei > 3, chromatin > 1, normal > 2, mitoses > 1)
       H₂ = 3 of (thickness > 6, size > 1, shape > 1, epithelial > 1,
                  nuclei > 8, normal > 9)
       H₃ = not used
```

As suggested by the rules, we used a thermometer (cumulative) coding of the nominally valued attributes so that less-than or greater-than subexpressions could be efficiently represented in the hidden weights. Such a representation is often useful in diagnosis tasks. We also limited the hidden weights to positive values due to the nature of the attributes.

| Problem | Network Topology | Training Set | | | Test Set | | |
|---------|------------------|--------------|---|---|----------|---|---|
| | | # of Patterns | Perf. of Network | Perf. of Rules | # of Patterns | Perf. of Network | Perf. of Rules |
| promoters | 228–0–1 | 105 | 100.0 % | 95.9 % | 1 | 94.2 % | 87.6 % |
| house-votes-84 | 16–0–1 | 387 | 97.3 % | 96.2 % | 43 | 95.7 % | 95.9 % |
| breast-cancer-wisc | 81–3–1 | 630 | 98.5 % | 96.3 % | 70 | 95.8 % | 95.2 % |

**Table 3: Simulation summary for UCI repository problems**

Taken as a whole our simulation results are encouraging, and we are conducting further research on rule extraction for more complex tasks.

## 5   CONCLUSION

We have described a general approach for extracting various types of $n$-of-$m$ symbolic rules from trained networks of sigmoidal units, assuming approximately Boolean activation behavior. While other methods for interpretation of this sort exist, ours represents a valuable price/performance point, offering easily-understood rules and good extraction performance with computational complexity that scales well with the expressiveness desired. The basic principles behind our approach may be flexibly applied to a wide variety of problems.

## Footnotes

[1] In fact the nesting may continue beyond one level. Thus sigmoidal units can compute expressions like $OR[AND(C_{n\text{-of-}m},\ C_{AND}),\ C_{OR}\ ]$. We have not yet experimented with extensions of this sort.

[2] All results in this paper are for networks trained using batch-mode back propagation on the cross-entropy error function. Training was stopped when outputs were within 0.05 of their target values for each pattern or a fixed number of epochs (typically 1000) was reached. Where indicated, a penalty term for non-Boolean hidden activations or hidden weight decay was added to the main error function. For the breast cancer problem shown in Table 4.3, hidden rules were extracted first and the output units were retrained briefly before extracting their rules. Results for the problems in Table 4.3 used leave-one-out testing or 10-fold cross-validation (with 10 different initial orderings) as indicated. All results are averages over 10 replications with different initial weights.

## References

[1] Alexander, J. A. (1994). *Template-based procedures for neural network interpretation.* MS Thesis. Department of Computer Science, University of Colorado, Boulder, CO.

[2] Andrews, R., Diederich, J., and Tickle, A. B. (1995). *A survey and critique of techniques for extracting rules from trained artificial neural networks.* To appear in Fu, L. M. (Ed.), Knowledge-Based Systems, Special Issue on Knowledge-Based Neural Networks.

[3] Mangasarian, O. L. and Wolberg, W. H. (1990). *Cancer diagnosis via linear programming.* SIAM News 23:5, pages 1 & 18.

[4] McMillan, C. (1992). *Rule induction in a neural network through integrated symbolic and subsymbolic processing.* PhD Thesis. Department of Computer Science, University of Colorado, Boulder, CO.

[5] Murphy, P. M. and Aha, D. W. (1994). *UCI repository of machine learning databases.* [Machine-readable data repository]. Irvine, CA: University of California, Department of Information and Computer Science. Monks data courtesy of Sebastian Thrun, promoters data courtesy of M. Noordewier and J. Shavlik, congressional voting data courtesy of Jeff Schlimmer, breast cancer data courtesy of Dr. William H. Wolberg (see also [3] above).

[6] Rumelhart, D. E., Hinton, G. E., and Williams, R. J. (1986). Learning internal representations by error propagation. In Rumelhart, D. E., McClelland, J. L., and the PDP Research Group, *Parallel Distributed Processing: Explorations in the Microstructure of Cognition. Volume 1: Foundations,* pages 318-362. Cambridge, MA: MIT Press.

[7] Thrun, S. B., and 23 other authors (1991). *The MONK's problems - A performance comparison of different learning algorithms.* Technical Report CS-CMU-91-197. Carnegie Mellon University, Pittsburgh, PA.

[8] Towell, G. and Shavlik, J. W. (1992). Interpretation of artificial neural networks: Mapping knowledge-based neural networks into rules. In Moody, J. E., Hanson, S. J., and Lippmann, R. P. (Eds.), *Advances in Neural Information Processing Systems,* 4:977-984. San Mateo, CA: Morgan Kaufmann.